# Confusion-Based Online Learning and a Passive-Aggressive Scheme

**Liva Ralaivola**
QARMA, Laboratoire d'Informatique Fondamentale de Marseille
Aix-Marseille University, France
`liva.ralaivola@lif.univ-mrs.fr`

## Abstract

This paper provides the first —to the best of our knowledge— analysis of online learning algorithms for multiclass problems when the *confusion* matrix is taken as a performance measure. The work builds upon recent and elegant results on noncommutative concentration inequalities, i.e. concentration inequalities that apply to matrices, and, more precisely, to matrix martingales. We do establish generalization bounds for online learning algorithms and show how the theoretical study motivates the proposition of a new confusion-friendly learning procedure. This learning algorithm, called `COPA` (for COnfusion Passive-Aggressive) is a passive-aggressive learning algorithm; it is shown that the update equations for `COPA` can be computed analytically and, henceforth, there is no need to recourse to any optimization package to implement it.

## 1 Introduction

This paper aims at promoting an infrequent way to tackle multiclass prediction problems: we advocate for the use of the confusion matrix —the matrix which reports the probability of predicting class $q$ for an instance of class $p$ for all potential label pair $(p, q)$— as *the* objective 'function' to be optimized. This way, we step aside the more widespread viewpoint of relying on the misclassification rate —the probability of misclassifying a point— as a performance measure for multiclass predictors. There are obvious reasons for taking this perspective, among which we may name the following. First, the confusion information is a finer-grain information than the misclassification rate, as it allows one to precisely identify the types of error made by a classifier. Second, the confusion matrix is independent of the class distributions, since it reports conditional probability distributions: a consequence is that a predictor learned to achieve a 'small' confusion matrix will probably be insensitive to class imbalance and it will also be robust to changes in class prior distributions between train and test data. Finally, there are many application domains such as medicine, bioinformatics, information retrieval, where the confusion matrix (or an estimate thereof) is precisely the object of interest for an expert who wants to assess the relevance of an automatic prediction procedure.

**Contributions.** We essentially provide two contributions. On the one hand, we provide a statistical analysis for the generalization ability of online learning algorithms producing predictors that aim at *optimizing the confusion*. This requires us to introduce relevant statistical quantities that are taken advantage of *via* a concentration inequality for matrix martingales proposed by [8]. Motivated by our statistical analysis, we propose an online learning algorithm from the family of *passive aggressive* learning algorithms [2]: this algorithm is inherently designed to optimize the confusion matrix and numerical simulations are provided that show it reaches its goal.

**Outline of the paper.** Section 2 formalizes our pursued objective of targeting a small confusion error. Section 3 provides our results regarding the ability of online confusion-aware learning

procedures to achieve a small confusion together with the update equations for COPA, a new passive-aggressive learning procedure designed to control the confusion risk. Section 4 reports numerical results that should be viewed as a proof of concept for the effectiveness of our algorithm.

## 2 Problem and Motivation

### 2.1 Notation

We focus on the problem of multiclass classification: the input space is denoted by $\mathcal{X}$ and the target space is $\mathcal{Y} = \{1, \ldots, Q\}$. The training sequence $\boldsymbol{Z} = \{Z_t = (X_t, Y_t)\}_{t=1}^T$ is made of $T$ identically and independently random pairs $Z_t = (X_t, Y_t)$ distributed according to some unknown distribution $D$ over $\mathcal{X} \times \mathcal{Y}$. The sequence of input data will be referred to as $\boldsymbol{X} = \{X_t\}_{t=1}^T$ and the sequence of corresponding labels $\boldsymbol{Y} = \{Y_t\}_{t=1}^T$. We may write that $\boldsymbol{Z}$ is distributed according to $D^T = \otimes_{t=1}^T D$. $\boldsymbol{Z}_{1:t}$ denotes the subsequence $\boldsymbol{Z}_{1:t} = \{(X_\tau, Y_\tau)\}_{\tau=1}^t$. We use $D_{X|y}$ for the conditional distribution on $X$ given that $Y = y$; therefore, for a given sequence $\boldsymbol{y} = (y_1, \ldots, y_T) \in \mathcal{Y}^T$, $D_{\boldsymbol{X}|\boldsymbol{y}} = \otimes_{t=1}^T D_{X|y_t}$ is the distribution of the random sample $\boldsymbol{X} = \{X_1, \ldots, X_T\}$ over $\mathcal{X}^T$ such that $X_t$ is distributed according to $D_{X|y_t}$. $\mathsf{E}[\cdot]$ and $\mathsf{E}_{X|y}$ respectively denote the expectation with respect to $D$ and $D_{X|y}$.

For a sequence $\boldsymbol{y}$ of labels, $\boldsymbol{T}(\boldsymbol{y}) = [T_1(\boldsymbol{y}) \cdots T_Q(\boldsymbol{y})] \in \mathbb{N}^Q$ is such that $T_q(\boldsymbol{y})$ is the number of times label $q$ appears in $\boldsymbol{y}$. Often, we will drop the dependence upon $\boldsymbol{y}$ for $\boldsymbol{T}(\boldsymbol{y})$. Throughout, we make the simplifying assumption that $T_q > 1$ for all $q$ —otherwise, our analysis still holds but extra care and notation must be taken for handling classes absent from $\boldsymbol{Z}$.

The space of hypotheses we consider is $\mathcal{H}$ (e.g. $\mathcal{H} \subseteq \{f : f : \mathcal{X} \to \mathbb{R}\}^Q$), and $\mathcal{A}$ designates an online learning algorithm that produces hypothesis $h_t \in \mathcal{H}$ when it encounters a new example $Z_t$.

Finally, $\boldsymbol{\ell} = (\ell_{q|p})_{1 \leq p,q \leq Q}$ is a family of *class-dependent* loss functionals $\ell_{q|p} : \mathcal{H} \times \mathcal{X} \to \mathbb{R}_+$. For a point $x \in \mathcal{X}$ of class $y \in \mathcal{Y}$, $\ell_{q|y}(h, x)$ is the cost of $h$'s favoring class $q$ over $y$ for $x$.

*Example* 1 (Misclassification Loss). The family of *(cost-sensitive) misclassification losses* $\boldsymbol{\ell}^{\text{misclass}}$ is defined as

$$\ell_{q|y}^{\text{misclass}}(h, x) \doteq \chi_{[h(x)=q]} C_{yq}, \tag{1}$$

where $C_{pq} \in \mathbb{R}_+$, $\forall p, q \in \mathcal{Y}$ and $\chi_{[E]} = 1$ if $E$ is true and $0$ otherwise.

*Example* 2 (Hinge Loss). The family of *multiclass hinge losses* $\boldsymbol{\ell}^{\text{hinge}}$ is such that, given $W = \{w_1, \ldots, w_Q\} \in \mathcal{X}^Q$ with $\sum w_q = \boldsymbol{0}$ and hypothesis $h_W$ such that $h_W(x) = [\langle w_1, x \rangle \cdots \langle w_Q, x \rangle]$

$$\ell_{q|y}^{\text{hinge}}(h_W, x) \doteq \left| \langle w_q, x \rangle + \frac{1}{Q-1} \right|_+, \quad \text{where } |\cdot|_+ = \max(0, \cdot). \tag{2}$$

### 2.2 From the Confusion Matrix to the Confusion Risk, and their Minimization

**Confusion matrix.** In many situations, e.g. class-imbalanced datasets, it is important not to measure the quality of a predictor $h$ based upon its classification rate

$$R(h) \doteq \mathbb{P}_{XY}(h(X) \neq Y) \tag{3}$$

only; this may lead to erroneous conclusions regarding the quality of $h$. Indeed, if, for instance, some class $q$ is predominantly present in the data at hand, say $\mathbb{P}(Y = q) = 1 - \varepsilon$, for some small $\varepsilon > 0$, then the predictor $h_{\text{maj}}$ that always outputs $h_{\text{maj}}(x) = q$ regardless of $x$ has a classification error lower that is at most $\varepsilon$, whereas it *never* correctly predicts the class of data from classes $p \neq q$.

A more informative object is the *confusion matrix* $\mathcal{C}(h) \in \mathbb{R}^{Q \times Q}$ of $h$, which is defined as:

$$\mathcal{C}(h) \doteq (\mathbb{P}(h(X) = q)|Y = p)_{1 \leq p,q \leq Q}. \tag{4}$$

The nondiagonal entries of $\mathcal{C}(h)$ provides the information as to the types of confusion, and their prevalence, $h$ makes when predicting the class of some $x$. Let us now abuse the notation and denote $\mathcal{C}(h)$ the confusion matrix where the diagonal entries are zeroed. It is straightforward to see, that, if $\boldsymbol{\pi} = [\mathbb{P}(Y = 1) \cdots \mathbb{P}(Y = Q)]$ is the vector of class prior distributions then, the misclassification rate $R(h)$ (cf. (3)) can be retrieved as

$$R(h) = \|\boldsymbol{\pi}\mathcal{C}(h)\|_1,$$

where $\|\cdot\|_1$ denotes the 1-norm. This says that, with little additional information, the misclassification rate might be obtained from the confusion matrix, while the converse is not true.

Is is clear that having the confusion matrix $\mathcal{C}(h)$ be zero means that $h$ is a perfect predictor. When such situation is not possible (if the data are corrupted by noise, for instance), a valuable objective might be to look for a classifier $h$ having a confusion matrix as close to zero as possible. Here, we choose to measure the closeness to zero of matrices through the operator norm $\|\cdot\|$, defined as:

$$\|M\| \doteq \max_{\boldsymbol{v} \neq \boldsymbol{0}} \frac{\|M\boldsymbol{v}\|_2}{\|\boldsymbol{v}\|_2}, \tag{5}$$

where $\|\cdot\|_2$ is the standard Euclidean norm —$\|M\|$ is merely the largest singular value of $M$. In addition to being a valid norm, the operator norm has the following nice property, that will be of help to us. Given two matrices $A = (a_{ij})$ and $B = (b_{ij})$ of the same dimensions

$$0 \leq a_{ij} \leq b_{ij}, \ \forall i, j \Rightarrow \|A\| \leq \|B\|. \tag{6}$$

Given the equivalence between norms and the definition of the operator norm, it is easy to see that

$$R(h) \leq \sqrt{Q}\|\mathcal{C}(h)\|,$$

and targetting a small confusion matrix for $h$ may have the consequence of implying a small misclassification risk $R(h)$.

The discussion conducted so far brings us to a natural goal of multiclass learning, that of *minimizing the norm of the confusion matrix,* i.e. that of solving the following optimization problem

$$\min_{h \in \mathcal{H}} \|\mathcal{C}(h)\|.$$

However, addressing this question directly poses a number of problems, both from the statistical and algorithmic sides: a) it is not possible to compute $\mathcal{C}(h)$ as it depends on the unknown distribution $D$ and b) relying on empirical estimates of $\mathcal{C}(h)$ as is folk in statistical learning, requires to deal with the indicator function $\chi_{[]}$ that appears in (1), which is not optimization-friendly.

**Confusion Risk.**   In order to deal with the latter problem and prepare the ground for tackling the former one from a theoretical point of view, we now introduce and discuss the *confusion risk*, which is parameterized by a family of loss functions that may be surrogate for the indicator loss $\chi_{[]}$.

**Definition 1** (Confusion Risk)**.**  The *confusion risk* $\mathcal{C}_{\boldsymbol{\ell}}(h)$ of $h$ is defined as

$$\mathcal{C}_{\boldsymbol{\ell}}(h) = \left( c_{pq}^{\boldsymbol{\ell}} \right)_{1 \leq p, q \leq Q} \in \mathbb{R}^{Q \times Q}, \ \text{with } c_{pq}^{\boldsymbol{\ell}} \doteq \begin{cases} \mathsf{E}_{X|p}\ell_{q|p}(h, X) & \text{if } p \neq q, \\ 0 & \text{otherwise.} \end{cases} \tag{7}$$

Observe that if the family $\boldsymbol{\ell}^{\mathrm{misclass}}$ of losses from Example 1 is retained, and $C_{pq} = 1$ for all $p, q$ then $\mathcal{C}_{\boldsymbol{\ell}^{\mathrm{misclass}}}(h)$ is precisely the confusion matrix discussed above (with the diagonal set to zero).

Similarly as before, the $\boldsymbol{\ell}$-risk $R_{\boldsymbol{\ell}}(h)$ is defined as $R_{\boldsymbol{\ell}}(h) \doteq \|\boldsymbol{\pi}\mathcal{C}_{\boldsymbol{\ell}}(h)\|_1$ and $R_{\boldsymbol{\ell}}(h) \leq \sqrt{Q}\|\mathcal{C}_{\boldsymbol{\ell}(h)}\|$.

The following lemma directly comes from Equation (6).

**Lemma 1.** *Let $h \in \mathcal{H}$. If $0 \leq \chi_{[h(x) \neq p]} \leq \ell_{q|p}(h, x)$, $\forall x \in \mathcal{X}$, $\forall p, q \in \mathcal{Y}$, then $\|\mathcal{C}(h)\| \leq \|\mathcal{C}_{\boldsymbol{\ell}}(h)\|$.*

This says that if we recourse to surrogate $\boldsymbol{\ell}$ that upper bounds the 0-1 indicator function then the norm of the confusion risk is always larger than the norm of the confusion matrix.

Armed with the confusion risk and the last lemma, we may now turn towards the legitimate objective

$$\min_{h \in \mathcal{H}} \|\mathcal{C}_{\boldsymbol{\ell}}(h)\|,$$

a small value of $\|\mathcal{C}_{\boldsymbol{\ell}}(h)\|$ implying a small value of $\|\mathcal{C}(h)\|$ (which was our primary goal).

It is still impossible to solve this problem because of the unknown distribution $D$ according to which expectations are computed. However, as already evoked, it is possible to derive learning strategies based on empirical estimates of the confusion risk, that ensures $\|\mathcal{C}_{\ell}(h)\|$ will be small. The next section is devoted to show how this could be done.

# 3 Bounds on the Confusion Risk via Online Learning and COPA

## 3.1 (Empirical) Online Confusion Risk

Assume an online learning algorithm $\mathcal{A}$ that outputs hypotheses from a family $\mathcal{H}$: run on the traning sequence $\boldsymbol{Z} \sim D^T$, $\mathcal{A}$ outputs hypotheses $\mathbf{h} = \{h_0, \ldots, h_T\}$, where $h_0$ is an arbitrary hypothesis.

**Definition 2** ($\widehat{\mathcal{L}}_{|p}(\cdot, \cdot)$ and $\mathcal{L}_{|p}(\cdot)$ matrices). *Let $\boldsymbol{\ell} = (\ell_{q|p})$ be a family of losses and let $p \in \mathcal{Y}$. For $h \in \mathcal{H}$ and $x \in \mathcal{X}$, we define*

$$\widehat{\mathcal{L}}_{|p}(h, x) = (\widehat{l}_{uv})_{1 \leq u, v \leq Q} \in \mathbb{R}^{Q \times Q}, \quad \text{with } \widehat{l}_{uv} \doteq \begin{cases} \ell_{v|u}(h, x) & \text{if } u = p \text{ and } v \neq u, \\ 0 & \text{otherwise.} \end{cases} \tag{8}$$

The matrix $\mathcal{L}_{|p}(h) \in \mathbb{R}^{Q \times Q}$ is naturally given by

$$\mathcal{L}_{|p}(h) = \mathsf{E}_{X|p} \widehat{\mathcal{L}}_{|p}(h, X). \tag{9}$$

*Remark* 1. Note that the only row that may be nonzero in $\widehat{\mathcal{L}}_{|p}(h, x)$ and $\mathcal{L}_{|p}(h)$ is the $p$th row.

**Definition 3** ((Conditional) Online Confusion Risk). *Let $\boldsymbol{y} \in \mathcal{Y}^T$ be a fixed sequence of labels and $\boldsymbol{Y}$ a random sequence of $T$ labels. Let $\mathbf{h} = \{h_0, \ldots, h_{T-1}\}$ be a sequence of $T$ hypotheses.*

The conditional and non-conditional confusion risks $\mathcal{C}_{\boldsymbol{\ell}, \boldsymbol{y}}(\mathbf{h})$ and $\mathcal{C}_{\boldsymbol{\ell}, \boldsymbol{Y}}(\mathbf{h})$ are defined as

$$\mathcal{C}_{\boldsymbol{\ell}, \boldsymbol{y}}(\mathbf{h}) \doteq \sum_{t=1}^T \frac{1}{T_{y_t}} \mathcal{L}_{|y_t}(h_{t-1}), \quad \text{and,} \quad \mathcal{C}_{\boldsymbol{\ell}, \boldsymbol{Y}}(\mathbf{h}) \doteq \sum_{t=1}^T \frac{1}{T_{Y_t}} \mathcal{L}_{|Y_t}(h_{t-1}). \tag{10}$$

*Remark* 2. $\mathcal{C}_{\boldsymbol{\ell}, \boldsymbol{Y}}(\mathbf{h})$ is a random variable. In order to provide our generalization bounds, it will be more convenient for us to work with the conditional confusion $\mathcal{C}_{\boldsymbol{\ell}, \boldsymbol{y}}(\mathbf{h})$. A simple argument will then allow us to provide generalization bounds for $\mathcal{C}_{\boldsymbol{\ell}, \boldsymbol{Y}}(\mathbf{h})$.

**Definition 4.** *Let $\boldsymbol{y} \in \mathcal{Y}^T$ be a fixed sequence of labels. Let $\mathbf{h} = \{h_0, \ldots, h_{T-1}\}$ be the hypotheses output by $\mathcal{A}$ when run on $\boldsymbol{Z}^{\boldsymbol{y}} = \{(X_t, y_t)\}_{t=1}^T$, such that $X_t$ is distributed according to $D_{X|y_t}$.*

The (non-)conditional empirical online confusion risks $\widehat{\mathcal{C}}_{\boldsymbol{\ell}, \boldsymbol{y}}(\mathbf{h}, \boldsymbol{X})$ and $\widehat{\mathcal{C}}_{\boldsymbol{\ell}, \boldsymbol{Y}}(\mathbf{h}, \boldsymbol{X})$ are

$$\widehat{\mathcal{C}}_{\boldsymbol{\ell}, \boldsymbol{y}}(\mathbf{h}, \boldsymbol{X}) \doteq \sum_{t=1}^T \frac{1}{T_{y_t}} \widehat{\mathcal{L}}_{|y_t}(h_{t-1}, X_t), \quad \text{and,} \quad \widehat{\mathcal{C}}_{\boldsymbol{\ell}, \boldsymbol{Y}}(\mathbf{h}, \boldsymbol{X}) \doteq \sum_{t=1}^T \frac{1}{T_{Y_t}} \widehat{\mathcal{L}}_{|Y_t}(h_{t-1}, X_t). \tag{11}$$

We now are almost ready to provide our results. We just need to introduce a pivotal result that is a corollary of Theorem 7.1 from [8], a proof of which can be found in the appendix.

**Corollary 1** (Rectangular Matrix Azuma). *Consider a sequence $\{U_k\}$ of $d_1 \times d_2$ random matrices, and a fixed sequence of scalars $\{M_k\}$ such that*

$$\mathsf{E}_{U_k|U_1 \ldots U_{k-1}} U_k = \mathbf{0}, \text{ and } \|U_k\| \leq M_k \text{ almost surely.}$$

*Then, for all $t > 0$,*

$$\mathbb{P}\left\{\left\|\sum_k U_k\right\| \geq t\right\} \leq (d1 + d2) \exp\left\{-\frac{t^2}{2\sigma^2}\right\}, \quad \text{with } \sigma^2 \doteq \sum_k M_k^2.$$

## 3.2 New Results

This subsection reports our main results.

**Theorem 1.** *Suppose that the losses in $\boldsymbol{\ell}$ are such that $0 \leq \ell_{q|p} \leq M$ for some $M > 0$. Fix $\boldsymbol{y} \in \mathcal{Y}^T$. For any $\delta \in (0; 1]$, it holds with probability $1 - \delta$ over the draw of $\boldsymbol{X} \sim D_{\boldsymbol{X}|\boldsymbol{y}}$ that*

$$\left\|\mathcal{C}_{\boldsymbol{\ell}, \boldsymbol{y}}(\mathbf{h}) - \widehat{\mathcal{C}}_{\boldsymbol{\ell}, \boldsymbol{y}}(\mathbf{h}, \boldsymbol{X})\right\| \leq M\sqrt{2Q \sum_p \frac{1}{T_p} \log \frac{Q}{\delta}} \tag{12}$$

*where $\mathbf{h} = \{h_0, \ldots, h_{T-1}\}$ is the set of hypotheses output by $\mathcal{A}$ when provided with $\{(X_t, y_t)\}_{t=1}^T$.*

*Therefore, with probability at least $1 - \delta$*

$$\|\mathcal{C}_{\boldsymbol{\ell},\boldsymbol{y}}(\mathbf{h})\| \leq \left\|\widehat{\mathcal{C}}_{\boldsymbol{\ell},\boldsymbol{y}}(\mathbf{h}, \boldsymbol{X})\right\| + M\sqrt{2Q\sum_p \frac{1}{T_p}\log\frac{Q}{\delta}}. \tag{13}$$

*Proof.* The proof is straightforward using Corollary 1. Indeed, consider the random variable

$$U_t \doteq \frac{1}{T_{y_t}}\mathcal{L}_{|y_t}(h_{t-1}) - \frac{1}{T_{y_t}}\widehat{\mathcal{L}}_{|y_t}(h_{t-1}, X_t).$$

On the one hand, we observe that:

$$\mathsf{E}_{X_t|\boldsymbol{X}_{1:t-1},\boldsymbol{y}}U_t = \mathsf{E}_{X_t|\boldsymbol{X}_{1:t-1},\boldsymbol{y}_{1:t}}U_t = \mathbf{0},$$

since $\mathsf{E}_{X_t|\boldsymbol{X}_{1:t-1}\boldsymbol{y}_{1:t}}\widehat{\mathcal{L}}_{|y_t}(h_{t-1}, X_t) = \mathcal{L}_{|y_t}(h_{t-1})$. On the other hand, introducing

$$\Delta_{t,q} \doteq \mathsf{E}_{X_t|y_t}\ell_{q|y_t}(h_{t-1}, X_t) - \ell_{q|y_t}(h_{t-1}, X_t),$$

we observe that

$$\|U_t\| = \sup_{\boldsymbol{v}:\|\boldsymbol{v}\|\leq 1}\|U_t\boldsymbol{v}\|_2 = \frac{1}{T_{y_t}}\sup_{\boldsymbol{v}:\|\boldsymbol{v}\|\leq 1}\left|\sum_{q\neq y_t}\Delta_{t,q}v_q\right| = \frac{1}{T_{y_t}}\sqrt{\sum_{q\neq y_t}\Delta_{t,q}^2} \leq \frac{M}{T_{y_t}}\sqrt{Q}.$$

where we used that the only row of $U_t$ not to be zero is its $y_t$th row (see Remark 1), the fact that $\sup_{\boldsymbol{v}:\|\boldsymbol{v}\|\leq 1}\boldsymbol{v}\cdot\mathbf{u} = \|\mathbf{u}\|_2$ and the assumption $0 \leq \ell_{q|p} \leq M$, which gives that $|\Delta_{t,q}| \leq M$.

Using Corollary 1 on the matrix martingale $\{U_t\}$, where $\|U_t\| \leq M\sqrt{Q}/T_{y_t}$ almost surely, gives

$$\mathbb{P}\left\{\left\|\sum_t U_t\right\| \geq \varepsilon\right\} \leq 2Q\exp\left\{-\frac{\varepsilon^2}{2M^2Q\sum_t \frac{1}{T_{y_t}^2}}\right\}$$

Setting the right hand side to $\delta$ gives that, with probability at least $1 - \delta$

$$\left\|\sum_t U_t\right\| \leq M\sqrt{2Q\sum_t \frac{1}{T_{y_t}^2}\ln\frac{2Q}{\delta}}.$$

$\sum_t T_{y_t}^{-2} = \sum_p T_p^{-1}$ gives (12). The triangle inequality $|\|A\| - \|B\|| \leq \|A - B\|$ gives (13). $\qquad\square$

If one takes a little step back to fully understand Theorem 1, it may not be as rejoicing as expected. Indeed, it provides a bound on the norm of the average confusion risks of hypotheses $h_0, \ldots, h_{T-1}$, which, from a practical point of view, does not say much about the norm of the confusion risk of a specific hypothesis. In fact, as is usual in online learning [1], it provides a bound on the confusion risk of the Gibbs classifier, which uniformly samples over $h_0, \ldots, h_{T-1}$ before outputting a prediction. Just as emphasized by [1], things may turn a little bit more enticing when the loss functions $\ell$ are convex with respect to their first argument, i.e.

$$\forall h, h' \in \mathcal{H}, \forall p, q \in \mathcal{Y}, \forall\lambda \in [0,1], \; \ell_{q|p}(\lambda h + (1-\lambda)h', x) \leq \lambda\ell_{q|p}(h) + (1-\lambda)\ell_{q|p}(h', x). \tag{14}$$

In that case, we may show the following theorem, that is much more compatible with the stated goal of trying to find a hypothesis that has small (or at least, controlled) confusion risk.

**Theorem 2.** *In addition to the assumptions of Theorem 1 we now assume that $\boldsymbol{\ell}$ is made of convex losses (as defined in (14)).*

*For any $\delta \in (0; 1]$, it holds with probability $1 - \delta$ over the draw of $\boldsymbol{X} \sim D_{\boldsymbol{X}|\boldsymbol{y}}$ that*

$$\|\mathcal{C}_{\boldsymbol{\ell}}(\underline{h})\| \leq \left\|\widehat{\mathcal{C}}_{\boldsymbol{\ell},\boldsymbol{y}}(\mathbf{h}, \boldsymbol{X})\right\| + M\sqrt{2Q\sum_p \frac{1}{T_p}\log\frac{Q}{\delta}}, \quad \text{with } \underline{h} \doteq \frac{1}{T}\sum_{t=1}^T h_{t-1}. \tag{15}$$

*Proof.* It is a direct consequence of the convexity of $\boldsymbol{\ell}$ combined with Equation (6). $\qquad\square$

It is now time to give the argument allowing us to state results for the non-conditional online confusion risks. If a random event $\mathcal{E}(A, B)$ defined with respect to random variables $A$ and $B$ is such that $\mathbb{P}_{A|B=b}(\mathcal{E}(A, b)) \geq 1 - \delta$ for all possible values of $b$ then $\mathbb{P}_{AB}(\mathcal{E}(A, B)) = \sum_b \mathbb{P}_{A|B=b}(\mathcal{E}(A, b))\mathbb{P}_B(B = b) \geq \sum_b(1 - \delta)\mathbb{P}_B(B = b) = 1 - \delta$. The results of Theorem 1 and Theorem 2 may be therefore stated in terms of $\boldsymbol{Y}$ instead of $\boldsymbol{y}$.

In light of the generic results of this subsection, we are now ready to motivate and derive a new online learning algorithm that aims at a small confusion risk.

### 3.3 Online Learning with COPA

This subsection presents a new online algorithm, COPA (for COnfusion Passive-Aggressive learning). Before giving the full detail of our algorithm, we further discuss implications of Theorem 2.

A first message from Theorem 2 is that, provided the functions $\ell$ considered are convex, it is relevant to use the average hypothesis $\underline{h}$ as a predictor. We indeed know by (15) that the confusion risk of $\underline{h}$ is bounded by $\|\widehat{\mathcal{C}}_{\boldsymbol{\ell},\boldsymbol{y}}(\mathbf{h}, \boldsymbol{X})\|$, plus some quantity directly related to the number of training data encountered. The second message from the theorem is that the focus naturally comes to $\|\widehat{\mathcal{C}}_{\boldsymbol{\ell},\boldsymbol{y}}(\mathbf{h}, \boldsymbol{X})\|$ and the question as to how minimize this quantity.

According to Definition 4, $\widehat{\mathcal{C}}_{\boldsymbol{\ell},\boldsymbol{y}}(\mathbf{h}, \boldsymbol{X})$ is the sum of instantaneous confusion matrices $\widehat{\mathcal{L}}_{|y_t}(h_{t-1}, X_t)/T_{y_t}$. In light of (6), it does make sense to try to minimize each entry of $\widehat{\mathcal{L}}_{|y_t}(h, X_t)/T_{y_t}$ with respect to $h$ to get $h_t$, with the hope that the instantaneous risk of $h_t$ on $X_{t+1}$ will be small: one may want to minimize the norm of $\widehat{\mathcal{L}}_{|y_t}(h, X_t)/T_{y_t}$ and pose a problem like the following:

$$\min_h \left\| \frac{1}{T_{y_t}} \widehat{\mathcal{L}}_{|y_t}(h, X_t) \right\|.$$

However, as remarked before, $\widehat{\mathcal{L}}_{|y_t}$ has only one nonzero row, its $y_t$th. Therefore, the operator norm of $\widehat{\mathcal{L}}_{|y_t}(h, X_t)/T_{y_t}$ is simply the Euclidean norm of its $y_t$th row. Trying to minimize the square Euclidean norm of this row might be written as

$$\min_h \frac{1}{T_{y_t}^2} \sum_{q \neq y_t} \ell_{q|y_t}^2(h, X_t). \tag{16}$$

This last equation is the starting point of COPA. To see how the connection is made, we make some instantiations. The hypothesis space $\mathcal{H}$ is made of linear classifiers, so that a sequence of vectors $W = \{w_1, \ldots, w_Q\}$ with $\sum_q w_q = \mathbf{0}$ defines a hypothesis $h_W$. The family $\boldsymbol{\ell}$ COPA builds upon is

$$\ell_{q|p}(h_W, x) = \left| \langle w_q, x \rangle + \frac{1}{Q-1} \right|_+, \ \forall p, q \in \mathcal{Y}.$$

In other words, COPA is an instance of Example 2. We focus on this setting because it is known that, in the batch scenario, it provides Bayes consistent classifiers [3, 4]. Given a current set of vectors $W_t = \{w_1^t, \ldots, w_Q^t\}$, using (16), and implementing a passive-aggressive learning strategy [2], the new set of vectors $W_{t+1}$ is searched as the solution of

$$\min_{W, \sum_q w_q = \mathbf{0}} \frac{1}{2} \sum_{q=1}^Q \|w_q - w_q^t\|_2^2 + \frac{C}{2T_y^2} \sum_{q \neq y} \left| \langle w_q, x \rangle + \frac{1}{Q-1} \right|_+^2. \tag{17}$$

It turns out that it is possible to find the solution of this optimization problem without having to recourse to any involved optimization procedure. This is akin to a result obtained by [6], which applies for another family of loss functions. We indeed prove the following theorem (proof in supplementary material).

**Theorem 3** (COPA update). *Suppose we want to solve*

$$\min_{W, \sum_q w_q = \mathbf{0}} \frac{1}{2} \sum_{q=1}^Q \|w_q - w_q^t\|_2^2 + \frac{C}{2} \sum_{q \neq y} \left| \langle w_q, x \rangle + \frac{1}{Q-1} \right|_+^2. \tag{18}$$

**Algorithm 1** `COPA`

---

**Input:** $\boldsymbol{z} = \{(x_t, y_t)\}_{t=1}^{T}$ training set (realization of $\boldsymbol{Z}$), $R$ number of epochs over $\boldsymbol{z}$, $C$ cost
**Output:** $W = \{w_1, \ldots, w_Q\}$, the classification vectors

  $\tau = 0$
  $w_1^0 = \ldots = w_Q^0$
  **for** r=1 to R **do**
    **for** t=1 to T **do**
      receive $(x_t, y_t)$
      compute $\boldsymbol{\alpha}^*$ according to (20)
      $\forall q$, perform the update: $w_q^{\tau+1} \leftarrow w_q^{\tau} - \left( \alpha_q^* - \frac{1}{Q} \sum_{q=1}^{I^*} \alpha_q^* \right) x_t$
      $\tau \leftarrow \tau + 1$
    **end for**
  **end for**
  $\forall q, w_q \leftarrow \frac{1}{\tau} \sum_{k=1}^{\tau} w_q^k$

---

*Let $\ell_q^t$ be defined as*

$$\ell_q^t \doteq \langle w_q, x \rangle + 1/(Q-1).$$

*Let $\sigma$ be a permutation defined over $\{1, \ldots, Q-1\}$ taking values in $\mathcal{Y} \backslash \{y\}$ such that*

$$\ell_{\sigma(1)}^t \geq \ldots \geq \ell_{\sigma(Q-1)}^t.$$

*Let $I^*$ be the largest index $I \in \{1, \ldots, Q-1\}$ such that*

$$\ell_{\sigma(I)}^t + \frac{\|x\|^2}{\kappa Q - (I-1)\|x\|^2} \sum_{q=1}^{I-1} \ell_{\sigma(q)}^t > 0, \quad \text{with } \kappa \doteq \frac{1}{C} + \|x\|^2 \tag{19}$$

*If $\mathcal{I}^*$ is set to $\mathcal{I}^* \doteq \{\sigma(1), \ldots, \sigma(I^*)\}$, then we may define $\boldsymbol{\alpha}^* = [\alpha_1^* \cdots \alpha_Q^*]$ as*

$$\alpha_q^* \doteq \begin{cases} \frac{1}{\kappa} \left( \ell_q^t + \frac{\|x\|^2}{Q} s_\alpha(\mathcal{I}^*) \right) & \text{if } q \in \mathcal{I}^* \\ 0 & \text{otherwise} \end{cases} , \text{ where } s_\alpha(\mathcal{I}^*) \doteq \frac{Q}{\kappa Q - I^* \|x\|^2} \sum_{q \in \mathcal{I}^*} \ell_q^t. \tag{20}$$

*and the vectors*

$$w_q^* \doteq w_q^t - \left( \alpha_q^* - \frac{1}{Q} \sum_{q=1}^{I^*} \alpha_q^* \right) x, \ q = 1, \ldots, Q \tag{21}$$

*are the solution of optimization problem (18). These equations provide* `COPA`*'s update procedure.*

The full `COPA` algorithm is depicted in Algorithm 1.

## 4  Numerical Simulations

We here report results of preliminary simulations of `COPA` on a toy dataset. We generate a set of 5000 samples according to three Gaussian distributions each of variance $\sigma^2 \mathbb{I}$ with $\sigma = 0.5$. One of the Gaussian is centered at $(0, 1)$, the other at $(1, 0)$ and the last one at $(-1, 0)$. The respective weights of the Gaussian distributions are 0.9, 0.05 and 0.05. The first generated sample is used to choose the parameter $C$ of `COPA` with a half split of the data between train and test; 10 other samples of size 5000 are generated and split as 2500 for learning and 2500 for testing and the results are averaged on the 10 samples. We compare the results of `COPA` to that of a simple multiclass perceptron procedure (the number of epochs for each algorithm is set to 5). As recommended by the theory we average the classification vector to get our predictors (both for `COPA` and the perceptron).

The essential finding of these simulations is that `COPA` and the perceptron achieve similar rate of classification accuracy, 0.85 and 0.86, respectively but the norm of the confusion matrix of `COPA` is 0.10 and that of the Perceptron is 0.18. This means `COPA` indeed does its job in trying to minimize the confusion risk.

# 5 Conclusion

In this paper, we have provided new bounds for online learning algorithms aiming at controlling their confusion risk. To the best of our knowledge, these results are the first of this kind. Motivated by the theoretical results, we have proposed a passive-aggressive learning algorithm, COPA, that has the seducing property that its updates can be computed easily, without having to resort to any optimization package. Preliminary numerical simulations tend to support the effectiveness of COPA.

In addition to complementary numerical simulations, we want to pursue our work in several directions. First, we would like to know whether efficient update equation can be derived if a simple hinge, instead of a square hinge is used. Second, we would like to see if a full regret-style analysis can be made to study the properties of COPA. Also, we are interested in comparing the merits of our theoretical results with those recently proposed in [5] and [7], which propose to address learning with the confusion matrix from the algorithmic stability and the PAC-Bayesian viewpoints. Finally, we would like to see how the proposed material can be of some use in the realm of structure prediction and by extension, in the case where the confusion matrix to consider is infinite-dimensional.

**Acknowledgments.** Work partially supported by Pascal 2 NoE ICT-216886-NOE, French ANR Projects ASAP (ANR-09-DEFIS–001) and GRETA (ANR-12-BS02-0004).

# Appendix

**Theorem 4** (Matrix Azuma-Hoeffding, [8])**.** *Consider a finite sequence $\{X_k\}$ of self-adjoint matrices in dimension $d$, and a fixed sequence $\{A_k\}$ of self-adjoint matrices that satisfy $\mathsf{E}_{k-1} X_k = \mathbf{0}$ and*

$$X_k^2 \preccurlyeq A_k^2, \text{ and } A_k X_k = X_k A_k, \text{ almost surely.}$$

*Then, for all $t \geq 0$,*

$$\mathbb{P}\left\{ \lambda_{\max}\left( \sum_k X_k \right) \geq t \right\} \leq d \cdot e^{-t^2/2\sigma^2},$$

*with $\sigma^2 = \left\| \sum_k A_k^2 \right\|$.*

*Proof of Corollary 1.* To prove the result, it suffices to make use of the dilation technique and apply Theorem 4. The self-adjoint dilation $\mathcal{D}(U)$ of a matrix $U \in \mathbb{R}^{d_1 \times d_2}$ is the self-adjoint matrix $\mathcal{D}(U)$ of order $d_1 + d_2$ defined by

$$\mathcal{D}(U) = \left[ \begin{array}{cc} \mathbf{0} & U \\ U^* & \mathbf{0} \end{array} \right]$$

where $U^*$ is the adjoint of $U$ (as $U$ has only real coefficient here, $U^*$ is the transpose of $U$).

As recalled in [8], $\|\mathcal{D}(U)\| = \|U\|$ and, therefore, the largest eigenvalue $\lambda_{\max}$ of $\mathcal{D}^2(U)$ is equal to $\|U\|^2$ and $\mathcal{D}^2(U) \preccurlyeq \|U\|^2 \mathbb{I}$.

Considering $U_k$, we get that, almost surely:

$$\mathcal{D}^2(U_k) \preccurlyeq M_k^2 \mathbb{I},$$

and since dilation is a linear operator, we have that

$$\mathsf{E}_{U_k|U_1 \cdots U_{k-1}} \mathcal{D}(U_k) = \mathbf{0}.$$

The sequence of matrices $\{\mathcal{D}(U_k)\}$ is therefore a matrix martingale that verifies the assumption of Theorem 4, the application of which gives

$$\mathbb{P}\left\{ \lambda_{\max}\left( \sum_k \mathcal{D}(U_k) \right) \geq t \right\} \leq (d_1 + d_2) \exp\left\{ -\frac{t^2}{2\sigma^2} \right\},$$

with $\sigma^2 = \sum_k M_k^2$. Thanks to the linearity of $\mathcal{D}$,

$$\lambda_{\max}\left( \sum_k \mathcal{D}(U_k) \right) = \lambda_{\max}\left( \mathcal{D}\left( \sum_k U_k \right) \right) = \left\| \sum_k U_k \right\|,$$

which closes the proof. □

# References

[1] N. Cesa-Bianchi, A. Conconi, and C. Gentile. On the generalization ability of online learning algorithms. *IEEE Transactions on Information Theory*, 50(9):2050–2057, 2004.

[2] Koby Crammer, Ofer Dekel, Joseph Keshet, Shai Shalev-Shwartz, and Yoram Singer. Online passive-aggressive algorithms. *Journal of Machine Learning Research*, 7:551–585, 2006.

[3] Y. Lee. Multicategory support vector machines, theory, and application to the classification of microarray data and satellite radiance data. Technical report, University of Wisconsin, 2002.

[4] Y. Lee, Y. Lin, and G. Wahba. Multicategory support vector machines. *Journal of the American Statistical Association*, 99:67–81, march 2004.

[5] P. Machart and L. Ralaivola. Confusion matrix stability bounds for multiclass classification. Technical report, Aix-Marseille University, 2012.

[6] S. Matsushima, N. Shimizu, K. Yoshida, T. Ninomiya, and H. Nakagawa. Exact passive-aggressive algorithm for multiclass classification using support class. In *SDM 10*, pages 303–314, 2010.

[7] E. Morvant, S. Koço, and L. Ralaivola. PAC-Bayesian Generalization Bound on Confusion Matrix for Multi-Class Classification. In John Langford and Joelle Pineau, editors, *International Conference on Machine Learning*, pages 815–822, Edinburgh, United Kingdom, 2012.

[8] J. A. Tropp. User-friendly tail bounds for sums of random matrices. *Foundations of Computational Mathematics*, pages 1–46, 2011.

